# Particle Filtering for Nonparametric Bayesian Matrix Factorization

**Frank Wood**
Department of Computer Science
Brown University
Providence, RI 02912
fwood@cs.brown.edu

**Thomas L. Griffiths**
Department of Psychology
University of California, Berkeley
Berkeley, CA 94720
tom_griffiths@berkeley.edu

## Abstract

Many unsupervised learning problems can be expressed as a form of matrix factorization, reconstructing an observed data matrix as the product of two matrices of latent variables. A standard challenge in solving these problems is determining the dimensionality of the latent matrices. Nonparametric Bayesian matrix factorization is one way of dealing with this challenge, yielding a posterior distribution over possible factorizations of unbounded dimensionality. A drawback to this approach is that posterior estimation is typically done using Gibbs sampling, which can be slow for large problems and when conjugate priors cannot be used. As an alternative, we present a particle filter for posterior estimation in nonparametric Bayesian matrix factorization models. We illustrate this approach with two matrix factorization models and show favorable performance relative to Gibbs sampling.

## 1   Introduction

One of the goals of unsupervised learning is to discover the latent structure expressed in observed data. The nature of the learning problem will vary depending on the form of the data and the kind of latent structure it expresses, but many unsupervised learning problems can be viewed as a form of matrix factorization – i.e. decomposing an observed data matrix, $\mathbf{X}$, into the product of two or more matrices of latent variables. If $\mathbf{X}$ is an $N \times D$ matrix, where $N$ is the number of $D$-dimensional observations, the goal is to find a low-dimensional latent feature space capturing the variation in the observations making up $\mathbf{X}$. This can be done by assuming that $\mathbf{X} \approx \mathbf{Z}\mathbf{Y}$, where $\mathbf{Z}$ is a $N \times K$ matrix indicating which of (and perhaps the extent to which) $K$ latent features are expressed in each of the $N$ observations and $\mathbf{Y}$ is a $K \times D$ matrix indicating how those $K$ latent features are manifest in the $D$ dimensional observation space. Typically, $K$ is less than $D$, meaning that $\mathbf{Z}$ and $\mathbf{Y}$ provide an efficient summary of the structure of $\mathbf{X}$.

A standard problem for unsupervised learning algorithms based on matrix factorization is determining the dimensionality of the latent matrices, $K$. Nonparametric Bayesian statistics offers a way to address this problem: instead of specifying $K$ a priori and searching for a "best" factorization, nonparametric Bayesian matrix factorization approaches such as those in [1] and [2] estimate a posterior distribution over factorizations with unbounded dimensionality (i.e. letting $K \to \infty$). This remains computationally tractable because each model uses a prior that ensures that $\mathbf{Z}$ is sparse, based on the Indian Buffet Process (IBP) [1]. The search for the dimensionality of the latent feature matrices thus becomes a problem of posterior inference over the number of non-empty columns in $\mathbf{Z}$.

Previous work on nonparametric Bayesian matrix factorization has used Gibbs sampling for posterior estimation [1, 2]. Indeed, Gibbs sampling is the standard inference algorithm used in nonparametric Bayesian methods, most of which are based on the Dirichlet process [3, 4]. However, recent

work has suggested that sequential Monte Carlo methods such as particle filtering can provide an efficient alternative to Gibbs sampling in Dirichlet process mixture models [5, 6].

In this paper we develop a novel particle filtering algorithm for posterior estimation in matrix factorization models that use the IBP, and illustrate its applicability to two specific models – one with a conjugate prior, and the other without a conjugate prior but tractable in other ways. Our particle filtering algorithm is by nature an "on-line" procedure, where each row of $\mathbf{X}$ is processed only once, in sequence. This stands in comparison to Gibbs sampling, which must revisit each row many times to converge to a reasonable representation of the posterior distribution. We present simulation results showing that our particle filtering algorithm can be significantly more efficient than Gibbs sampling for each of the two models, and discuss its applicability to the broad class of nonparametric matrix factorization models based on the IBP.

## 2 Nonparametric Bayesian Matrix Factorization

Let $\mathbf{X}$ be an observed $N \times D$ matrix. Our goal is to find a representation of the structure expressed in this matrix in terms of the latent matrices $\mathbf{Z}$ ($N \times K$) and $\mathbf{Y}$ ($K \times D$). This can be formulated as a statistical problem if we view $\mathbf{X}$ as being produced by a probabilistic generative process, resulting in a probability distribution $P(\mathbf{X}|\mathbf{Z}, \mathbf{Y})$. The critical assumption necessary to make this a matrix factorization problem is that the distribution of $\mathbf{X}$ is conditionally dependent on $\mathbf{Z}$ and $\mathbf{Y}$ only through the product $\mathbf{ZY}$. Although defining $P(\mathbf{X}|\mathbf{Z}, \mathbf{Y})$ allows us to use methods such as maximum-likelihood estimation to find a point estimate, our goal is to instead compute a posterior distribution over possible values of $\mathbf{Z}$ and $\mathbf{Y}$. To do so we need to specify a prior over the latent matrices $P(\mathbf{Z}, \mathbf{Y})$, and then we can use Bayes' rule to find the posterior distribution over $\mathbf{Z}$ and $\mathbf{Y}$

$$P(\mathbf{Z}, \mathbf{Y}|\mathbf{X}) \quad \propto \quad P(\mathbf{X}|\mathbf{Z}, \mathbf{Y})P(\mathbf{Z}, \mathbf{Y}). \tag{1}$$

This constitutes Bayesian matrix factorization, but two problems remain: the choice of $K$, and the computational cost of estimating the posterior distribution.

Unlike standard matrix factorization methods that require an a priori choice of $K$, nonparametric Bayesian approaches allow us to estimate a posterior distribution over $\mathbf{Z}$ and $\mathbf{Y}$ where the size of these matrices is unbounded. The models we discuss in this paper place a prior on $\mathbf{Z}$ that gives each "left-ordered" binary matrix (see [1] for details) probability

$$P(\mathbf{Z}) = \frac{\alpha^{K_+}}{\prod_{h=1}^{2^N-1} K_h!}\exp\{-\alpha H_N\} \prod_{k=1}^{K_+} \frac{(N - m_k)!(m_k - 1)!}{N!} \tag{2}$$

where $K_+$ is the number of columns of $\mathbf{Z}$ with non-zero entries, $m_k$ is the number of 1's in column $k$, $N$ is the number of rows, $H_N = \sum_{i=1}^N 1/i$ is the $N^{\text{th}}$ harmonic number, and $K_h$ is the number of columns in $\mathbf{Z}$ that when read top-to-bottom form a sequence of 1's and 0's corresponding to the binary representation of the number $h$. This prior on $\mathbf{Z}$ is a distribution on sparse binary matrices that favors those that have few columns with many ones, with the rest of the columns being all zeros.

This distribution can be derived as the outcome of a sequential generative process called the *Indian buffet process* (IBP) [1]. Imagine an Indian restaurant into which $N$ customers arrive one by one and serve themselves from the buffet. The first customer loads her plate from the first Poisson($\alpha$) dishes. The $i^{\text{th}}$ customer chooses dishes proportional to their popularity, choosing a dish with probability $m_k/i$ where $m_k$ is the number of people who have choosen the $k^{\text{th}}$ dish previously, then chooses Poisson($\alpha/i$) new dishes. If we record the choices of each customer on one row of a matrix whose columns correspond to a dishes on the buffet (1 if chosen, 0 if not) then (the left-ordered form of) that matrix constitutes a draw from the distribution in Eqn. 2. The order in which the customers enter the restaurant has no bearing on the distribution of $\mathbf{Z}$ (up to permutation of the columns), making this distribution exchangeable.

In this work we assume that $\mathbf{Z}$ and $\mathbf{Y}$ are independent, with $P(\mathbf{Z}, \mathbf{Y}) = P(\mathbf{Z})P(\mathbf{Y})$. As shown in Fig. 1, since we use the IBP prior for $P(\mathbf{Z})$, $\mathbf{Y}$ is a matrix with an infinite number of rows and $D$ columns. We can take any appropriate distribution for $P(\mathbf{Y})$, and the infinite number of rows will not pose a problem because only $K_+$ rows will interact with non-zero elements of $\mathbf{Z}$. A posterior distribution over $\mathbf{Z}$ and $\mathbf{Y}$ implicitly defines a distribution over the effective dimensionality of these

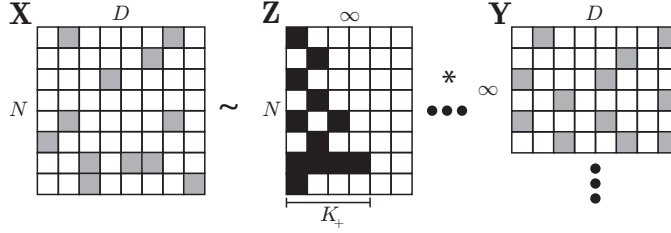

Figure 1: Nonparametric Bayesian matrix factorization. The data matrix $\mathbf{X}$ is the product of $\mathbf{Z}$ and $\mathbf{Y}$, which have an unbounded number of columns and rows respectively.

matrices, through $K_+$. This approach to nonparametric Bayesian matrix factorization has been used for both continuous [1, 7] and binary [2] data matrices $\mathbf{X}$.

Since the posterior distribution defined in Eqn. 1 is generally intractable, Gibbs sampling has previously been employed to construct a sample-based representation of this distribution. However, generally speaking, Gibbs sampling is slow, requiring each entry in $\mathbf{Z}$ and $\mathbf{Y}$ to be repeatedly updated conditioned on all of the others. This problem is compounded in contexts where the the number of rows of $\mathbf{X}$ increases as a consequence of new observations being introduced, where the Gibbs sampler would need to be restarted after the introduction of each new observation.

## 3 Particle Filter Posterior Estimation

Our approach addresses the problems faced by the Gibbs sampler by exploiting the fact that the prior on $\mathbf{Z}$ is recursively decomposable. To explain this we need to introduce new notation, let $\mathbf{X}^{(i)}$ be the $i^{\text{th}}$ row of $\mathbf{X}$, and $\mathbf{X}^{(1:i)}$ and $\mathbf{Z}^{(1:i)}$ be all the rows of $\mathbf{X}$ and $\mathbf{Z}$ up to $i$ respectively. Note that because the IBP prior is recursively decomposable it is easy to sample from $P(\mathbf{Z}^{(1:i)}|\mathbf{Z}^{(1:i-1)})$; to do so simply follow the IBP in choosing dishes for the $i^{\text{th}}$ customer given the record of which dishes were chosen by the first $i-1$ customers (see Algorithm 1). Applying Bayes' rule, we can write the posterior on $\mathbf{Z}^{(1:i)}$ and $\mathbf{Y}$ given $\mathbf{X}^{(1:i)}$ in the following form

$$P(\mathbf{Z}^{(1:i)}, \mathbf{Y}|\mathbf{X}^{(1:i)}) \propto P(\mathbf{X}^{(i)}|\mathbf{Z}^{(1:i)}, \mathbf{Y}, \mathbf{X}^{(1:i-1)})P(\mathbf{Z}^{(1:i)}, \mathbf{Y}|\mathbf{X}^{(1:i-1)}). \qquad (3)$$

Here we do not index $\mathbf{Y}$ as it is always an infinite matrix.[1]

If we could evaluate $P(\mathbf{Z}^{(1:i-1)}, \mathbf{Y}|\mathbf{X}^{(1:i-1)})$, we could obtain weighted samples (or "particles") from $P(\mathbf{Z}^{(1:i)}, \mathbf{Y}|\mathbf{X}^{(1:i)})$ using importance sampling with a proposal distribution of

$$P(\mathbf{Z}^{(1:i)}, \mathbf{Y}|\mathbf{X}^{(1:i-1)}) \;=\; \sum_{\mathbf{Z}^{(1:i-1)}} P(\mathbf{Z}^{(1:i)}|\mathbf{Z}^{(1:i-1)})P(\mathbf{Z}^{(1:i-1)}, \mathbf{Y}|\mathbf{X}^{(1:i-1)}) \qquad (4)$$

and taking

$$w_\ell \propto P(\mathbf{X}^{(i)}|\mathbf{Z}^{(1:i)}_{(\ell)}, \mathbf{Y}_{(\ell)}, \mathbf{X}^{(1:i-1)}) \qquad (5)$$

as the weight associated with the $\ell^{\text{th}}$ particle. However, we could also use a similar scheme to approximate $P(\mathbf{Z}^{(1:i-1)}, \mathbf{Y}|\mathbf{X}^{(1:i-1)})$ if we could evaluate $P(\mathbf{Z}^{(1:i-2)}, \mathbf{Y}|\mathbf{X}^{(1:i-2)})$. Following Eq. 4, we could then approximately generate a set of weighted particles from $P(\mathbf{Z}^{(1:i)}, \mathbf{Y}|\mathbf{X}^{(1:i-1)})$ by using the IBP to sample a value from $P(\mathbf{Z}^{(1:i)}|\mathbf{Z}^{(1:i-1)}_{(\ell)})$ for each particle from $P(\mathbf{Z}^{(1:i-1)}, \mathbf{Y}|\mathbf{X}^{(1:i-1)})$ and carrying forward the weights associated with those particles. This "particle filtering" procedure defines a recursive importance sampling scheme for the full posterior $P(\mathbf{Z}, \mathbf{Y}|\mathbf{X})$, and is known as sequential importance sampling [8]. When applied in its basic form this procedure can produce particles with extreme weights, so we resample the particles at each iteration of the recursion from the distribution given by their normalized weights and set $w_\ell = 1/L$ for all $\ell$, which is a standard method known as sequential importance resampling [8].

The procedure defined in the previous paragraphs is a general-purpose particle filter for matrix-factorization models based on the IBP. This procedure will work even when the prior defined on

**Algorithm 1** Sample $P(\mathbf{Z}^{(1:i)}|\mathbf{Z}^{(1:i-1)},\alpha)$ using the Indian Buffet process

1: $\mathbf{Z} \leftarrow \mathbf{Z}^{(1:i-1)}$
2: **if** $i = 1$ **then**
3:     sample $K_i^{\mathrm{new}} \sim \mathrm{Poisson}(\alpha)$
4:     $\mathbf{Z}_{i,1:K_i^{\mathrm{new}}} \leftarrow 1$
5: **else**
6:     $K_+ \leftarrow$ number of non-zero columns in $\mathbf{Z}$
7:     **for** $k = 1,\ldots,K_+$ **do**
8:         sample $z_{i,k}$ according to $P(z_{i,k} = 1) \sim \mathrm{Bernoulli}(\frac{m_{-i,k}}{i})$
9:     **end for**
10:    sample $K_i^{\mathrm{new}} \sim \mathrm{Poisson}(\frac{\alpha}{i})$
11:    $\mathbf{Z}_{i,K_++1:K_++K_i^{\mathrm{new}}} \leftarrow 1$
12: **end if**
13: $\mathbf{Z}^{(1:i)} \leftarrow \mathbf{Z}$

---

$\mathbf{Y}$ is not conjugate to the likelihood (and is much simpler than other algorithms for using the IBP with non-conjugate priors, e.g. [9]). However, the procedure can be simplified further in special cases. The following example applications illustrate the particle filtering approach for two different models. In the first case, the prior over $\mathbf{Y}$ is conjugate to the likelihood which means that $\mathbf{Y}$ need not be represented. In the other case, although the prior is not conjugate and thus $\mathbf{Y}$ does need to be explicitly represented, we present a way to improve the efficiency of this general particle filtering approach by taking advantage of certain analytic conditionals. The particle filtering approach results in significant improvements in performance over Gibbs sampling in both models.

## 4 A Conjugate Model: Infinite Linear-Gaussian Matrix Factorization

In this model, explained in detail in [1], the entries of both $\mathbf{X}$ and $\mathbf{Y}$ are continuous. We report results on the modeling of image data of the same kind as was originally used to demonstrate the model in [1]. Here each row of $\mathbf{X}$ is an image, each row of $\mathbf{Z}$ indicates the "latent features" present in that image, such as the objects it contains, and each column of $\mathbf{Y}$ indicates the pixel values associated with a latent feature.

The likelihood for this image model is matrix Gaussian

$$P(\mathbf{X}|\mathbf{Z},\mathbf{Y},\sigma_x) = \frac{1}{(2\pi\sigma_X^2)^{ND/2}}\exp\{-\frac{1}{2\sigma_X^2}\mathrm{tr}((\mathbf{X}-\mathbf{Z}\mathbf{Y})^T(\mathbf{X}-\mathbf{Z}\mathbf{Y}))\}$$

where $\sigma_X^2$ is the noise variance. The prior on the parameters of the latent features is also Gaussian

$$P(\mathbf{Y}|\sigma_Y) = \frac{1}{(2\pi\sigma_Y^2)^{KD/2}}\exp\{-\frac{1}{2\sigma_Y^2}\mathrm{tr}(\mathbf{Y}^T\mathbf{Y})\}$$

with each element having variance $\sigma_Y^2$. Because both the likelihood and the prior are matrix Gaussian, they form a conjugate pair and $\mathbf{Y}$ can be integrated out to yield the collapsed likelihood,

$$P(\mathbf{X}|\mathbf{Z},\sigma_x) = \frac{1}{(2\pi)^{ND/2}\sigma_X^{(N-K_+)D}\sigma_Y^{K+D}|\mathbf{Z}_+^T\mathbf{Z}_+ + \frac{\sigma_X^2}{\sigma_Y^2}\mathbf{I}_{K_+}|^{D/2}}\exp\{-\frac{1}{2\sigma_X^2}\mathrm{tr}(\mathbf{X}^T\boldsymbol{\Sigma}^{-1}\mathbf{X})\} \quad (6)$$

which is matrix Gaussian with covariance $\boldsymbol{\Sigma}^{-1} = \mathbf{I} - \mathbf{Z}_+(\mathbf{Z}_+^T\mathbf{Z} + \frac{\sigma_X^2}{\sigma_Y^2}\mathbf{I}_{K_+})^{-1}\mathbf{Z}_+^T$. Here $\mathbf{Z}_+ = \mathbf{Z}_{1:i,1:K_+}$ is the first $K_+$ columns of $\mathbf{Z}$ and $K_+$ is the number of non-zero columns of $\mathbf{Z}$.

### 4.1 Particle Filter

The use of a conjugate prior means that we do not need to represent $\mathbf{Y}$ explicitly in our particle filter. In this case the particle filter recursion shown in Eqns. 3 and 4 reduces to

$$P(\mathbf{Z}^{(1:i)}|\mathbf{X}^{(1:i)}) \propto P(\mathbf{X}^{(i)}|\mathbf{Z}^{(1:i)},\mathbf{X}^{(1:i-1)}) \sum_{\mathbf{Z}^{(1:i-1)}} P(\mathbf{Z}^{(1:i)}|\mathbf{Z}^{(1:i-1)})P(\mathbf{Z}^{(1:i-1)}|\mathbf{X}^{(1:i-1)})$$

and may be implemented as shown in Algorithm 2.

**Algorithm 2** Particle filter for Infinite Linear Gaussian Model

1: initialize $L$ particles $[\mathbf{Z}_\ell^{(0)}]$, $\ell = 1, \ldots, L$
2: **for** $i = 1, \ldots, N$ **do**
3:     **for** $\ell = 1, \ldots, L$ **do**
4:        sample $\mathbf{Z}_\ell^{(1:i)}$ from $\mathbf{Z}_\ell^{(1:i-1)}$ using Algorithm 1
5:        calculate $w_\ell$ using Eqns. 5 and 7
6:     **end for**
7:     normalize particle weights
8:     resample particles according to weight cumulative distribution
9: **end for**

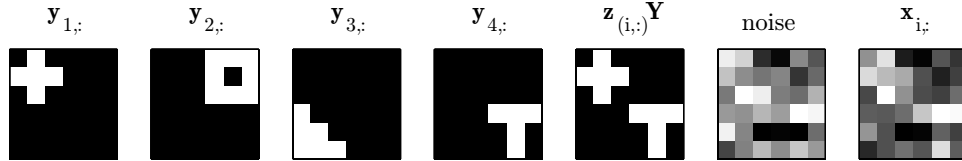

$$\mathbf{y}_{1,:} \qquad \mathbf{y}_{2,:} \qquad \mathbf{y}_{3,:} \qquad \mathbf{y}_{4,:} \qquad \mathbf{z}_{(i,:)}\mathbf{Y} \qquad \text{noise} \qquad \mathbf{x}_{i,:}$$

Figure 2: Generation of $\mathbf{X}$ under the linear Gaussian model. The first four images (left to right) correspond to the true latent features, i.e. rows of $\mathbf{Y}$. The fifth shows how the images get combined, with two source images added together by multiplying by a single row of $\mathbf{Z}$, $\mathbf{z}_{i,:} = [1\ 0\ 0\ 1]$. The sixth is Gaussian noise. The seventh image is the resulting row of $\mathbf{X}$.

Reweighting the particles requires computing $P(\mathbf{X}^{(i)}|\mathbf{Z}^{(1:i)}, \mathbf{X}^{(1:i-1)})$, the conditional probability of the most recent row of $\mathbf{X}$ given all the previous rows and $\mathbf{Z}$. Since $P(\mathbf{X}^{(1:i)}|\mathbf{Z}^{(1:i)})$ is matrix Gaussian we can find the required conditional distribution by following the standard rules for conditioning in Gaussians. Letting $\boldsymbol{\Sigma}_*^{-1} = \boldsymbol{\Sigma}^{-1}/\sigma_X^2$ be the covariance matrix for $\mathbf{X}^{(1:i)}$ given $\mathbf{Z}^{(1:i)}$, we can partition this matrix into four parts

$$\boldsymbol{\Sigma}_*^{-1} = \begin{bmatrix} \mathbf{A} & \vdots & \mathbf{c} \\ \cdots & \cdots & \cdots \\ \mathbf{c}^{\mathbf{T}} & \vdots & b \end{bmatrix}$$

where $\mathbf{A}$ is a matrix, $\mathbf{c}$ is a vector, and $b$ is a scalar. Then the conditional distribution of $\mathbf{X}^{(i)}$ is

$$\mathbf{X}^{(i)}|\mathbf{Z}^{(1:i)}, \mathbf{X}^{(1:i-1)} \sim \text{Gaussian}(\mathbf{c}^T\mathbf{A}^{-1}\mathbf{X}^{(1:i-1)}, b - \mathbf{c}^T\mathbf{A}^{-1}\mathbf{c}). \tag{7}$$

This requires inverting a matrix $\mathbf{A}$ which grows linearly with the size of the data; however, $\mathbf{A}$ is highly structured and this can be exploited to reduce the cost of this inversion [10].

## 4.2 Experiments

We compared the particle filter in Algorithm 2 with Gibbs sampling on an image dataset similar to that used in [1]. Due to space limitations we refer the reader to [1] for the details of the Gibbs sampler for this model. As illustrated in Fig. 2, our ground-truth $\mathbf{Y}$ consisted of four different $6 \times 6$ latent images. A $100 \times 4$ binary ground-truth matrix $\mathbf{Z}$ was generated with by sampling from $P(z_{i,k} = 1) = 0.5$. The observed matrix $\mathbf{X}$ was generated by adding Gaussian noise with $\sigma_X = 0.5$ to each entry of $\mathbf{Z}\mathbf{Y}$.

Fig. 3 compares results from the particle filter and Gibbs sampler for this model. The performance of the models was measured by comparing a general error metric computed over the posterior distributions estimated by each approach. The error metric (the vertical axis in Figs. 3 and 5) was computed by taking the expectation of the matrix $\mathbf{Z}\mathbf{Z}^T$ over the posterior samples produced by each algorithm and taking the summed absolute difference (i.e. $L_1$ norm) between the upper triangular portion of $E[\mathbf{Z}\mathbf{Z}^T]$ computed over the samples and the upper triangular portion of the true $\mathbf{Z}\mathbf{Z}^T$ (including the diagonal). See Fig. 4 for an illustration of the information conveyed by $\mathbf{Z}\mathbf{Z}^T$. This error metric measures the distance of the mean of the posterior to the ground-truth. It is zero if the mean of the distribution matches the ground truth. It grows as a function of the difference between the ground truth and the posterior mean, accounting both for any difference in the number of latent factors that are present in each observation and for any difference in the number of latent factors that are shared between all pairs of observations.

The particle filter was run using many different numbers of particles, $P$. For each value of $P$, the particle filter was run 10 times. The horizontal axis location of each errorbar in the plot is the mean

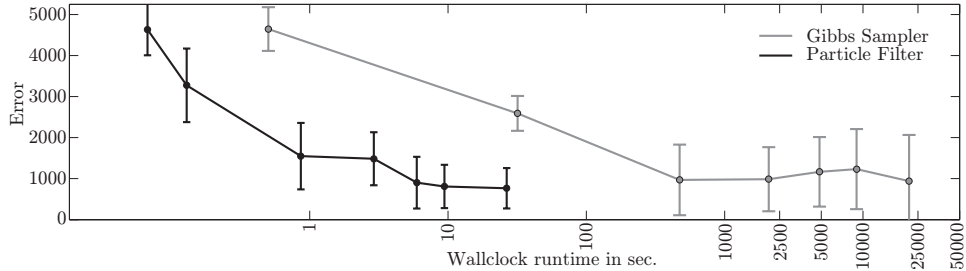

Figure 3: Performance results for particle filter vs. Gibbs sampling posterior estimation for the infinite linear Gaussian matrix factorization. Each point is an average over 10 runs with a particular number of particles or sweeps of the sampler $P = [1, 10, 100, 500, 1000, 2500, 5000]$ left to right, and error bars indicate the standard deviation of the error.

wall-clock computation time on 2 Ghz Athlon 64 processors running Matlab for the corresponding number of particles $P$ while the error bars indicate the standard deviation of the error. The Gibbs sampler was run for varying numbers of sweeps, with the initial $10\%$ of samples being discarded. The number of Gibbs sampler sweeps was varied and the results are displayed in the same way as described for the particle filter above. The results show that the particle filter attains low error in significantly less time than the Gibbs sampler, with the difference being an order or magnitude or more in most cases. This is a result of the fact that the particle filter considers only a single row of $\mathbf{X}$ on each iteration, reducing the cost of computing the likelihood.

## 5  A Semi-Conjugate Model: Infinite Binary Matrix Factorization

In this model, first presented in the context of learning hidden causal structure [2], the entries of both $\mathbf{X}$ and $\mathbf{Y}$ are binary. Each row of $\mathbf{X}$ represents the values of a single observed variable across $D$ trials or cases, each row of $\mathbf{Y}$ gives the values of a latent variable (a "hidden cause") across those trials or cases, and $\mathbf{Z}$ is the adjacency matrix of a bipartite Bayesian network indicating which latent variables influence which observed variables. Learning the hidden causal structure then corresponds to inferring $\mathbf{Z}$ and $\mathbf{Y}$ from $\mathbf{X}$. The model fits our schema for nonparametric Bayesian matrix factorization model (and hence is amenable to the use of our particle filter) since the likelihood function it uses depends only on the product $\mathbf{ZY}$.

The likelihood function for this model assumes that each entry of $\mathbf{X}$ is generated independently $P(\mathbf{X}|\mathbf{Z}, \mathbf{Y}) = \prod_{i,d} P(x_{i,d}|\mathbf{Z}, \mathbf{Y})$, with its probability given by the "noisy-OR" [11] of the causes that influence that variable (identified by the corresponding row of $\mathbf{Z}$) and are active for that case or trial (expressed in $\mathbf{Y}$). The probability that $x_{i,d}$ takes the value 1 is thus

$$P(x_{i,d} = 1|\mathbf{Z}, \mathbf{Y}) = 1 - (1 - \lambda)^{\mathbf{z}_{i,:} \cdot \mathbf{y}_{:,d}} (1 - \epsilon) \tag{8}$$

where $\mathbf{z}_{i,:}$ is the $i^{\text{th}}$ row of $\mathbf{Z}$, $\mathbf{y}_{:,d}$ is the $d^{\text{th}}$ column of $\mathbf{Y}$, and $\mathbf{z}_{i,:} \cdot \mathbf{y}_{:,d} = \sum_{k=1}^{K} z_{i,k} y_{k,d}$. The parameter $\epsilon$ sets the probability that $x_{i,d} = 1$ when no relevant causes are active, and $\lambda$ determines how this probability changes as the number of relevant active hidden causes increases. To complete the model, we assume that the entries of $\mathbf{Y}$ are generated independently from a Bernoulli process with parameter $p$, to give $P(\mathbf{Y}) = \prod_{k,d} p^{y_{k,d}} (1 - p)^{1 - y_{k,d}}$, and use the IBP prior for $\mathbf{Z}$.

### 5.1  Particle Filter

In this model the prior over $\mathbf{Y}$ is not conjugate to the likelihood, so we are forced to explicitly represent $\mathbf{Y}$ in our particle filter state, as outlined in Eqns. 3 and 4. However, we can define a more efficient algorithm than the basic particle filter due to the tractability of some integrals. This is why we call this model a "semi-conjugate" model.

The basic particle filter defined in Section 3 requires drawing the new rows of $\mathbf{Y}$ from the prior when we generate new columns of $\mathbf{Z}$. This can be problematic since the chance of producing an assignment of values to $\mathbf{Y}$ that has high probability under the likelihood can be quite low, in effect wasting many particles. However, if we can analytically marginalize out the new rows of $\mathbf{Y}$, we can avoid sampling those values from the prior and instead sample them from the posterior, in

**Algorithm 3** Particle filter for Infinite Binary Matrix Factorization

1: initialize $L$ particles $[\mathbf{Z}_\ell^{(0)}, \mathbf{Y}_\ell^{(0)}]$, $\ell = 1, \ldots, L$
2: **for** $i = 1, \ldots, N$ **do**
3:   **for** $\ell = 1, \ldots, L$ **do**
4:     sample $\mathbf{Z}_\ell^{(i)}$ from $\mathbf{Z}_\ell^{(i-1)}$ using Algorithm 1
5:     calculate $w_\ell$ using Eqns. 5 and 8
6:   **end for**
7:   normalize particle weights
8:   resample particles according to weight CDF
9:   **for** $\ell = 1, \ldots, L$ **do**
10:     sample $\mathbf{Y}_\ell^{(i)}$ from $P(\mathbf{Y}_\ell^{(i)}|\mathbf{Z}_\ell^{(1:i)}, \mathbf{Y}_\ell^{(1:i-1)}, \mathbf{X}^{(1:i)})$
11:   **end for**
12: **end for**

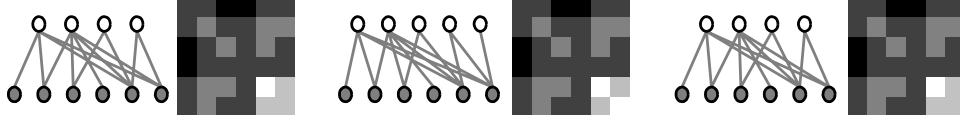

Figure 4: Infinite binary matrix factorization results. On the left is ground truth, the causal graph representation of $\mathbf{Z}$ and $\mathbf{Z}\mathbf{Z}^T$. The middle and right are particle filtering results; a single random particle $\mathbf{Z}$ and $E[\mathbf{Z}\mathbf{Z}^T]$ from a 500 and 10000 particle run middle and right respectively.

effect saving many of the potentially wasted particles. If we let $\mathbf{Y}^{(1:i)}$ denote the rows of $\mathbf{Y}$ that correspond to the first $i$ columns of $\mathbf{Z}$ and $\mathbf{Y}^{(i)}$ denote the rows (potentially more than 1) of $\mathbf{Y}$ that are introduced to match the new columns appearing in $\mathbf{Z}^{(i)}$, then we can write

$$P(\mathbf{Z}^{(1:i)}, \mathbf{Y}^{(1:i)}|\mathbf{X}^{(1:i)}) = P(\mathbf{Y}^{(i)}|\mathbf{Z}^{(1:i)}, \mathbf{Y}^{(1:i-1)}, \mathbf{X}^{(1:i)})P(\mathbf{Z}^{(1:i)}, \mathbf{Y}^{(1:i-1)}|\mathbf{X}^{(1:i)}) \quad (9)$$

where

$$P(\mathbf{Z}^{(1:i)}, \mathbf{Y}^{(1:i-1)}|\mathbf{X}^{(1:i)}) \propto P(\mathbf{X}^{(i)}|\mathbf{Z}^{(1:i)}, \mathbf{Y}^{(1:i-1)}, \mathbf{X}^{(1:i-1)})P(\mathbf{Z}^{(1:i)}, \mathbf{Y}^{(1:i-1)}|\mathbf{X}^{(1:i-1)}).$$
$$(10)$$

Thus, we can use the particle filter to estimate $P(\mathbf{Z}^{(1:i)}, \mathbf{Y}^{(1:i-1)}|\mathbf{X}^{(1:i)})$ (vs. $P(\mathbf{Z}^{(1:i)}, \mathbf{Y}^{(1:i)}|\mathbf{X}^{(1:i)})$) provided that we can find a way to compute $P(\mathbf{X}^{(i)}|\mathbf{Z}^{(1:i)}, \mathbf{Y}^{(1:i-1)})$ and sample from the distribution $P(\mathbf{Y}^{(i)}|\mathbf{Z}^{(1:i)}, \mathbf{Y}^{(1:i-1)}, \mathbf{X}^{(1:i)})$ to complete our particles.

The procedure described in the previous paragraph is possible in this model because, while our prior on $\mathbf{Y}$ is not conjugate to the likelihood, it is still possible to compute $P(\mathbf{X}^{(i)}|\mathbf{Z}^{(1:i)}, \mathbf{Y}^{(1:i-1)})$. The entries of $\mathbf{X}^{(i)}$ are independent given $\mathbf{Z}^{(1:i)}$ and $\mathbf{Y}^{(i)}$. Since the entries in each column of $\mathbf{Y}^{(i)}$ will influence only a single entry in $\mathbf{X}^{(i)}$, this independence is maintained when we sum out $\mathbf{Y}^{(i)}$. So we can derive an analytic solution to $P(\mathbf{X}^{(i)}|\mathbf{Z}^{(1:i)}, \mathbf{Y}^{(1:i-1)}) = \prod_d P(x_{i,d}|\mathbf{Z}^{(1:i)}, \mathbf{Y}^{(1:i-1)})$ where

$$P(x_{i,d} = 1|\mathbf{Z}^{(1:i)}, \mathbf{Y}^{(1:i-1)}) = 1 - (1-\epsilon)(1-\lambda)^\eta (1-\lambda p)^{K_i^{\text{new}}} \quad (11)$$

with $K_i^{\text{new}}$ being the number of new columns in $\mathbf{Z}^{(i)}$, and $\eta = \mathbf{z}_{i,1:K_+^{(1:i)}} \cdot \mathbf{y}_{1:K_+^{(1:i)},d}$. For a detailed derivation see [2]. This gives us the likelihood we need for reweighting particles $\mathbf{Z}^{(1:i)}$ and $\mathbf{Y}^{(1:i-1)}$. The posterior distribution on $\mathbf{Y}^{(i)}$ is straightforward to compute by combining the likelihood in Eqn. 8 with the prior $P(\mathbf{Y})$. The particle filtering algorithm for this model is given in Algorithm 3.

## 5.2 Experiments

We compared the particle filter in Algorithm 3 with Gibbs sampling on a dataset generated from the model described above, using the same Gibbs sampling algorithm and data generation procedure as developed in [2]. We took $K_+ = 4$ and $N = 6$, running the IBP multiple times with $\alpha = 3$ until a matrix $\mathbf{Z}$ of correct dimensionality ($6 \times 4$) was produced. This matrix is shown in Fig. 4 as a bipartite graph, where the observed variables are shaded. A $4 \times 250$ random matrix $\mathbf{Y}$ was generated with $p = 0.1$. The observed matrix $\mathbf{X}$ was then sampled from Eqn. 8 with parameters $\lambda = .9$ and $\epsilon = .01$. Comparison of the particle filter and Gibbs sampling was done using the procedure outlined in Section 4.2, producing similar results: the particle filter gave a better approximation to the posterior distribution in less time, as shown in Fig. 5.

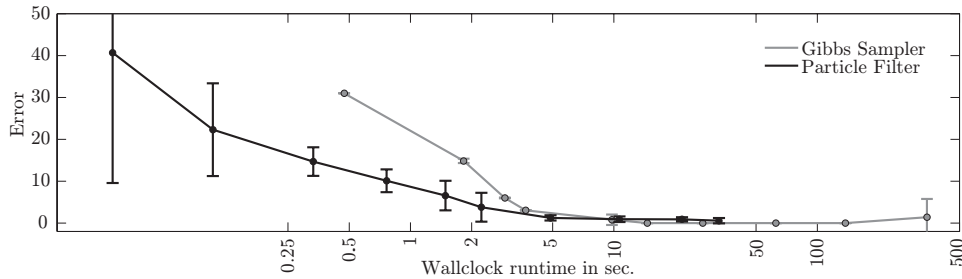

Figure 5: Performance results for particle filter vs. Gibbs sampling posterior estimation for the infinite binary matrix factorization model. Each point is an average over 10 runs with a particular number of particles or sweeps of the sampler $P = [1, 2, 5, 10, 20, 50, 100, 200, 500, 1000]$ from left to right, and error bars indicate the standard deviation of the error.

## 6  Conclusion

In this paper we have introduced particle filter posterior estimation for non-parametric Bayesian matrix factorization models based on the Indian buffet process. This approach is applicable to any Bayesian matrix factorization model with a sparse recursively decomposable prior. We have applied this approach with two different models, one with a conjugate prior and one with a non-conjugate prior, finding significant computational savings over Gibbs sampling for each. However, more work needs to be done to explore the strengths and weaknesses of these algorithms. In particular, simple sequential importance resampling is known to break down when applied to datasets with many observations, although we are optimistic that methods for addressing this problem that have been developed for Dirichlet process mixture models (e.g., [5]) will also be applicable in this setting. By exploring the strengths and weaknesses of different methods for approximate inference in these models, we hope to come closer to our ultimate goal of making nonparametric Bayesian matrix factorization into a tool that can be applied on the scale of real world problems.

**Acknowledgements** This work was supported by both NIH-NINDS R01 NS 50967-01 as part of the NSF/NIH Collaborative Research in Computational Neuroscience Program and NSF grant 0631518.

## Footnotes

[1]In practice, we need only keep track of the rows of $\mathbf{Y}$ that correspond to the non-empty columns of $\mathbf{Z}$, as the posterior distribution for the remaining entries is just the prior. Thus, if new non-empty columns are added in moving from $\mathbf{Z}^{(i-1)}$ to $\mathbf{Z}^{(i)}$, we need to expand the number of rows of $\mathbf{Y}$ that we represent accordingly.

## References

[1] T. L. Griffiths and Z. Ghahramani, "Infinite latent feature models and the Indian buffet process," Gatsby Computational Neuroscience Unit, Tech. Rep. 2005-001, 2005.

[2] F. Wood, T. L. Griffiths, and Z. Ghahramani, "A non-parametric Bayesian method for inferring hidden causes," in *Proceeding of the 22nd Conference on Uncertainty in Artificial Intelligence*.  in press, 2006.

[3] T. Ferguson, "A Bayesian analysis of some nonparametric problems," *The Annals of Statistics*, vol. 1, pp. 209–230, 1973.

[4] R. M. Neal, "Markov chain sampling methods for Dirichlet process mixture models," Department of Statistics, University of Toronto, Tech. Rep. 9815, 1998.

[5] P. Fearnhead, "Particle filters for mixture models with an unknown number of components," *Journal of Statistics and Computing*, vol. 14, pp. 11–21, 2004.

[6] S. N. MacEachern, M. Clyde, and J. Liu, "Sequential importance sampling for nonparametric Bayes models: the next generation," *The Canadian Journal of Statistics*, vol. 27, pp. 251–267, 1999.

[7] T. Griffiths and Z. Ghahramani, "Infinite latent feature models and the Indian buffet process," in *Advances in Neural Information Processing Systems 18*, Y. Weiss, B. Schölkopf, and J. Platt, Eds.  Cambridge, MA: MIT Press, 2006.

[8] A. Doucet, N. de Freitas, and N. Gordon, *Sequential Monte Carlo Methods in Practice*.  Springer, 2001.

[9] D. Görür, F. Jäkel, and C. R. Rasmussen, "A choice model with infinitely many latent features," in *Proceeding of the 23rd International Conference on Machine Learning*, 2006.

[10] S. Barnett, *Matrix Methods for Engineers and Scientists*.  McGraw-Hill, 1979.

[11] J. Pearl, *Probabilistic reasoning in intelligent systems*.  San Francisco, CA: Morgan Kaufmann, 1988.